# Convolutional-Recursive Deep Learning for 3D Object Classification

**Richard Socher, Brody Huval, Bharath Bhat, Christopher D. Manning, Andrew Y. Ng**
Computer Science Department, Stanford University, Stanford, CA 94305, USA
richard@socher.org, {brodyh,bbhat,manning}@stanford.edu, ang@cs.stanford.edu

## Abstract

Recent advances in 3D sensing technologies make it possible to easily record color and depth images which together can improve object recognition. Most current methods rely on very well-designed features for this new 3D modality. We introduce a model based on a combination of convolutional and recursive neural networks (CNN and RNN) for learning features and classifying RGB-D images. The CNN layer learns low-level translationally invariant features which are then given as inputs to multiple, fixed-tree RNNs in order to compose higher order features. RNNs can be seen as combining convolution and pooling into one efficient, hierarchical operation. Our main result is that even RNNs with random weights compose powerful features. Our model obtains state of the art performance on a standard RGB-D object dataset while being more accurate and faster during training and testing than comparable architectures such as two-layer CNNs.

## 1 Introduction

Object recognition is one of the hardest problems in computer vision and important for making robots useful in home environments. New sensing technology, such as the Kinect, that can record high quality RGB and depth images (RGB-D) has now become affordable and could be combined with standard vision systems in household robots. The depth modality provides useful extra information to the complex problem of general object detection [1] since depth information is invariant to lighting or color variations, provides geometrical cues and allows better separation from the background. Most recent methods for object recognition with RGB-D images use hand-designed features such as SIFT for 2d images [2], Spin Images [3] for 3D point clouds, or specific color, shape and geometry features [4, 5].

In this paper, we introduce the first convolutional-recursive deep learning model for object recognition that can learn from raw RGB-D images. Compared to other recent 3D feature learning methods [6, 7], our approach is fast, does not need additional input channels such as surface normals and obtains state of the art results on the task of detecting household objects. Fig. 1 outlines our approach. Code for training and testing is available at www.socher.org.

Our model starts with raw RGB and depth images and first separately extracts features from them. Each modality is first given to a single convolutional neural net layer (CNN, [8]) which provides useful translational invariance of low level features such as edges and allows parts of an object to be deformable to some extent. The pooled filter responses are then given to a recursive neural network (RNN, [9]) which can learn compositional features and part interactions. RNNs hierarchically project inputs into a lower dimensional space through multiple layers with tied weights and nonlinearities.

We also explore new deep learning architectures for computer vision. Our previous work on RNNs in natural language processing and computer vision [9, 10] (i) used a different tree structure for each input, (ii) employed a single RNN with one set of weights, (iii) restricted tree structures to be strictly

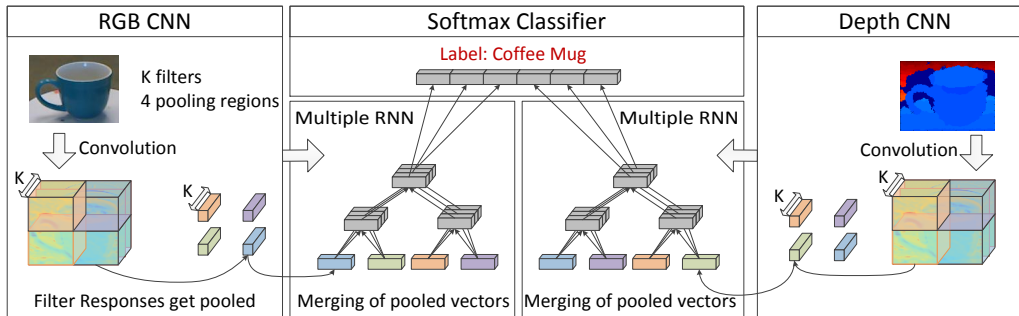

| RGB CNN | Softmax Classifier | Depth CNN |

Figure 1: An overview of our model: A single CNN layer extracts low level features from RGB and depth images. Both representations are given as input to a set of RNNs with random weights. Each of the many RNNs (around 100 for each modality) then recursively maps the features into a lower dimensional space. The concatenation of all the resulting vectors forms the final feature vector for a softmax classifier.

binary, and (iv) trained the RNN with backpropagation through structure [11, 12]. In this paper, we expand the space of possible RNN-based architectures in these four dimensions by using fixed tree structures and multiple RNNs on the same input and allow n-ary trees. We show that because of the CNN layer, fixing the tree structure does not hurt performance and it allows us to speed up recognition. Similar to recent work [13, 14] we show that performance of RNN models can improve with an increasing number of features. The hierarchically composed RNN features of each modality are concatenated and given to a joint softmax classifier.

Most importantly, we demonstrate that RNNs with random weights can also produce high quality features. So far random weights have only been shown to work for convolutional neural networks [15, 16]. Because the supervised training reduces to optimizing the weights of the final softmax classifier, a large set of RNN architectures can quickly be explored. By combining the above ideas we obtain a state of the art system for classifying 3D objects which is very fast to train and highly parallelizable at test time.

We first briefly describe the unsupervised learning of filter weights and their convolution to obtain low level features. Next we give details of how multiple random RNNs can be used to obtain high level features of the entire image. Then, we discuss related work. In our experiments we show quantitative comparisons of different models, analyze model ablations and describe our state-of-the-art results on the RGB-D dataset of Lai et al. [2].

## 2 Convolutional-Recursive Neural Networks

In this section, we describe our new CNN-RNN model. We first learn the CNN filters in an unsupervised way by clustering random patches and then feed these patches into a CNN layer. The resulting low-level, translationally invariant features are given to recursive neural networks. RNNs compose higher order features that can then be used to classify the images.

### 2.1 Unsupervised Pre-training of CNN Filters

We follow the procedure described by Coates et al. [13] to learn filters which will be used in the convolution. First, random patches are extracted into two sets, one for each modality (RGB and depth). Each set of patches is then normalized and whitened. The pre-processed patches are clustered by simply running $k$-means. Fig. 2 shows the resulting filters for both modalities. They capture standard edge and color features. One interesting result when applying this method to the depth channel is that the edges are much sharper. This is due to the large discontinuities between object boundaries and the background. While the depth channel is often quite noisy most of the features are still smooth.

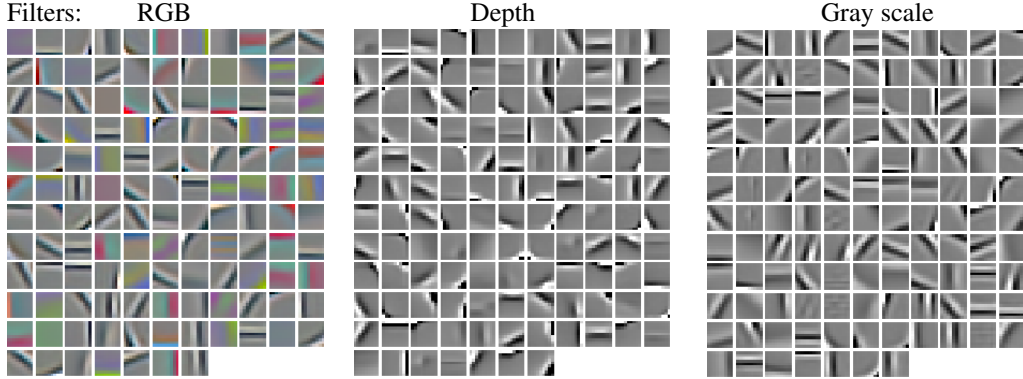

Figure 2: Visualization of the $k$-means filters used in the CNN layer after unsupervised pre-training: (**left**) Standard RGB filters (best viewed in color) capture edges and colors. When the method is applied to depth images (**center**) the resulting filters have sharper edges which arise due to the strong discontinuities at object boundaries. The same is true, though to a lesser extent, when compared to filters trained on gray scale versions of the color images (**right**).

## 2.2 A Single CNN Layer

To generate features for the RNN layer, a CNN architecture is chosen for its translational invariance properties. The main idea of CNNs is to convolve filters over the input image in order to extract features. Our single layer CNN is similar to the one proposed by Jarrett et. al [17] and consists of a convolution, followed by rectification and local contrast normalization (LCN). LCN was inspired by computational neuroscience and is used to contrast features within a feature map, as well as across feature maps at the same spatial location [17, 18, 14].

We convolve each image of size (height and width) $d_I$ with $K$ square filters of size $d_P$, resulting in $K$ filter responses, each of dimensionality $d_I - d_P + 1$. We then average pool them with square regions of size $d_\ell$ and a stride size of $s$, to obtain a pooled response with width and height equal to $r = (d_I - d_\ell)/s + 1$. So the output $X$ of the CNN layer applied to one image is a $K \times r \times r$ dimensional 3D matrix. We apply this same procedure to both color and depth images separately.

## 2.3 Fixed-Tree Recursive Neural Networks

The idea of recursive neural networks [19, 9] is to learn hierarchical feature representations by applying the same neural network recursively in a tree structure. In our case, the leaf nodes of the tree are $K$-dimensional vectors (the result of the CNN pooling over an image patch repeated for all $K$ filters) and there are $r^2$ of them.

In our previous RNN work [9, 10, 20] the tree structure depended on the input. While this allows for more flexibility, we found that for the task of object classification in conjunction with a CNN layer it was not necessary for obtaining high performance. Furthermore, the search over optimal trees slows down the method considerably as one can not easily parallelize the search or make use of parallelization of large matrix products. The latter could benefit immensely from new multicore hardware such as GPUs. In this work, we focus on fixed-trees which we can design to be balanced. Previous work also only combined pairs of vectors. We generalize our RNN architecture to allow each layer to merge blocks of adjacent vectors instead of only pairs.

We start with a 3D matrix $X \in \mathbb{R}^{K \times r \times r}$ for each image (the columns are $K$-dimensional). We define a block to be a list of adjacent column vectors which are merged into a parent vector $p \in \mathbb{R}^K$. In the following we use only square blocks for convenience. Blocks are of size $K \times b \times b$. For instance, if we merge vectors in a block with $b = 3$, we get a total size $128 \times 3 \times 3$ and a resulting list of vectors $(x_1, \ldots, x_9)$. In general, we have $b^2$ many vectors in each block. The neural network

for computing the parent vector is

$$p = f\left(W\begin{bmatrix}x_1 \\ \vdots \\ x_{b^2}\end{bmatrix}\right), \tag{1}$$

where the parameter matrix $W \in \mathbb{R}^{K \times b^2 K}$, $f$ is a nonlinearity such as $\tanh$. We omit the bias term which turns out to have no effect in the experiments below. Eq. 1 will be applied to all blocks of vectors in $X$ with the same weights $W$. Generally, there will be $(r/b)^2$ many parent vectors $p$, forming a new matrix $P_1$. The vectors in $P_1$ will again be merged in blocks just as those in matrix $X$ using Eq. 1 with the same tied weights resulting in matrix $P_2$. This procedure continues until only one parent vector remains. Fig. 3 shows an example of a pooled CNN output of size $K \times 4 \times 4$ and a RNN tree structure with blocks of 4 children.

The model so far has been unsupervised. However, our original task is to classify each block into one of many object categories. Therefore, we use the top vector $P_{top}$ as the feature vector to a softmax classifier. In order to minimize the cross entropy error of the softmax, we could backpropagate through the recursive neural network [12] and convolutional layers [8]. In practice, this is very slow and we will discuss alternatives in the next section.

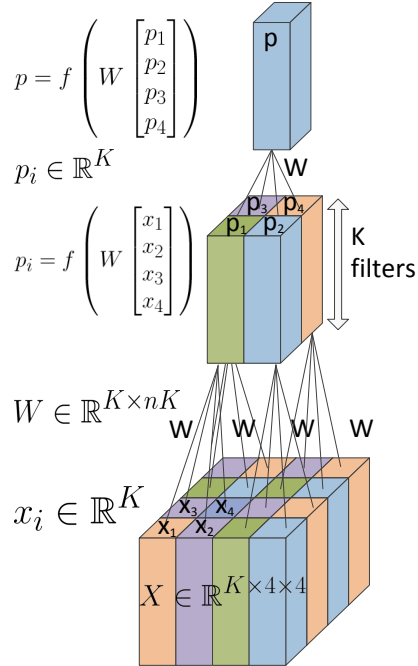

### 2.4 Multiple Random RNNs

Previous work used only a single RNN. We can actually use the 3D matrix $X$ as input to a number of RNNs. Each of $N$ RNNs will output a $K$-dimensional vector. After we forward propagate through all the RNNs, we concatenate their outputs to a $NK$-dimensional vector which is then given to the softmax classifier.

Instead of taking derivatives of the $W$ matrices of the RNNs which would require backprop through structure [11], we found that even RNNs with random weights produce high quality feature vectors. Similar results have been found for random weights in the closely related CNNs [16]. Before comparing to other approaches, we briefly review related work.

Figure 3: Recursive Neural Network applied to blocks: At each node, the same neural network is used to compute the parent vector of a set of child vectors. The original input matrix is the output of a pooled convolution.

## 3 Related Work

There has been great interest in object recognition and scene understanding using RGB-D data. Silberman and Fergus have published a 3D dataset for full scene understanding [21]. Koppula et al. also recently provided a new dataset for indoor scene segmentation [4].

The most common approach today for standard object recognition is to use well-designed features based on orientation histograms such as SIFT, SURF [22] or textons and give them as input to a classifier such as a random forest. Despite their success, they have several shortcomings such as being only applicable to one modality (grey scale images in the case of SIFT), not adapting easily to new modalities such as RGB-D or to varying image domains. There have been some attempts to modify these features to colored images via color histograms [23] or simply extending SIFT to the depth channel [2]. More advanced methods that generalize these ideas and can combine several important RGB-D image characteristics such as size, 3D shape and depth edges are kernel descriptors [5].

Another related line of work is about spatial pyramids in object classification, in particular the pyramid matching kernel [24]. The similarity is mostly in that our model also learns a hierarchical image representation that can be used to classify objects.

Another solution to the above mentioned problems is to employ unsupervised feature learning methods [25, 26, 27] (among many others) which have made large improvements in object recognition. While many deep learning methods exist for learning features from rgb images, few deep learning architectures have yet been investigated for 3D images. Very recently, Blum et al. [6] introduced convolutional $k$-means descriptors (CKM) for RGB-D data. They use SURF interest points and learn features using $k$-means similar to [28]. Their work is similar to ours in that they also learn features in an unsupervised way.

Very recent work by Bo et al. [7] uses unsupervised feature learning based on sparse coding to learn dictionaries from 8 different channels including grayscale intensity, RGB, depth scalars, and surface normals. Features are then used in hierarchical matching pursuit which consists of two layers. Each layer has three modules: batch orthogonal matching pursuit, pyramid max pooling, and contrast normalization. This results in a very large feature vector size of 188,300 dimensions which is used for classification.

Lastly, recursive autoencoders have been introduced by Pollack [19] and Socher et al. [10] to which we compare quantitatively in our experiment section. Recursive neural networks have been applied to full scene segmentation [9] but they used hand-designed features. Farabet et al. [29] also introduce a model for scene segmentation that is based on multi-scale convolutional neural networks and learns feature representations.

## 4    Experiments

All our experiments are carried out on the recent RGB-D dataset of Lai et al. [2]. There are 51 different classes of household objects and 300 instances of these classes. Each object instance is imaged from 3 different angles resulting in roughly 600 images per instance. The dataset consists of a total of 207,920 RGB-D images. We subsample every 5th frame of the 600 images resulting in a total of 120 images per instance.

In this work we focus on the problem of category recognition and we use the same setup as [2] and the 10 random splits they provide. All development is carried out on a separate split and model ablations are run on one of the 10 splits. For each split's test set we sample one object from each class resulting in 51 test objects, each with about 120 independently classified images. This leaves about 34,000 images for training our model. Before the images are given to the CNN they are resized to be $d_I = 148$.

Unsupervised pre-training for CNN filters is performed for all experiments by using $k$-means on 500,000 image patches randomly sampled from each split's training set. Before unsupervised pre-training, the $9 \times 9 \times 3$ patches for RGB and $9 \times 9$ patches for depth are individually normalized by subtracting the mean and divided by the standard deviation of its elements. In addition, ZCA whitening is performed to de-correlate pixels and get rid of redundant features in raw images [30]. A valid convolution is performed with filter bank size $K = 128$ and filter width and height of 9. Average pooling is then performed with pooling regions of size $d_\ell = 10$ and stride size $s = 5$ to produce a 3D matrix of size $128 \times 27 \times 27$ for each image.

Each RNN has non-overlapping child sizes of $3 \times 3$ applied spatially. This leads to the following matrices at each depth of the tree: $X \in \mathbb{R}^{128 \times 27 \times 27}$ to $P_1 \in \mathbb{R}^{128 \times 9 \times 9}$ to $P_2 \in \mathbb{R}^{128 \times 3 \times 3}$ to finally $P_3 \in \mathbb{R}^{128}$. We use 128 randomly initialized RNNs in both modalities. The combination of RGB and depth is done by concatenating the final features which have $2 \times 128^2 = 32,768$ dimensions.

### 4.1    Comparison to Other Methods

In this section we compare our model to related models in the literature. Table 1 lists the main accuracy numbers and compares to the published results [2, 5, 6, 7]. Recent work by Bo et al. [5] investigates multiple kernel descriptors on top of various features, including 3D shape, physical size of the object, depth edges, gradients, kernel PCA, local binary patterns,etc. In contrast, all our features are learned in an unsupervised way from the raw color and depth images. Blum et al. [6]

| Classifier | Extra Features for 3D;RGB | 3D | RGB | Both |
|---|---|---|---|---|
| Linear SVM [2] | Spin Images, efficient match kernel (EMK), random Fourier sets, width, depth, height; SIFT, EMK, texton histogram, color histogram | 53.1±1.7 | 74.3±3.3 | 81.9±2.8 |
| Kernel SVM [2] | same as above | 64.7±2.2 | 74.5±3.1 | 83.9±3.5 |
| Random Forest [2] | same as above | 66.8±2.5 | 74.7±3.6 | 79.6±4.0 |
| SVM [5] | 3D shape, physical size of the object, depth edges, gradients, kernel PCA, local binary patterns,multiple depth kernels | 78.8±2.7 | 77.7±1.9 | 86.2±2.1 |
| CKM [6] | SURF interest points | – | – | 86.4±2.3 |
| SP+HMP [7] | surface normals | 81.2±2.3 | 82.4±3.1 | 87.5±2.9 |
| CNN-RNN | – | 78.9±3.8 | 80.8±4.2 | 86.8±3.3 |

Table 1: Comparison of our CNN-RNN to multiple related approaches. We outperform all approaches except that of Bo et al. which uses an extra input modality of surface normals.

also learn feature descriptors and apply them sparsely to interest points. We outperform all methods except that of Bo et al. [7] who perform 0.7% better with a final feature vector that requires five times the amount of memory compared to ours. They make additional use of surface normals and gray scale images on top of RGB and depth channels and also learn features from these inputs with unsupervised methods based on sparse coding. Sparse coding is known to not scale well in terms of speed for large input dimensions [31].

## 4.2 Model Analysis

We analyze our model through several ablations and model variations. We picked one of the splits as our development fold and focus on RGB images and RNNs with random weights only unless otherwise noted.

**Two layer CNN**. Fig. 4 (left) shows a comparison between our CNN-RNN model and a two layer CNN. We compare a previously recommended architecture for CNNs [17] and one which uses filters trained with $k$-means. In both settings, the CNN-RNN outperforms the two layer CNN. Because it also requires many fewer matrix multiplication, it is approximately $4\times$ faster in our experiments compared to a second CNN layer. However, the main bottleneck of our method is still the first CNN layer. Both models could benefit from fast GPU implementations [32, 33].

**Tree structured neural nets with untied weights**. Fig. 4 (left) also gives results when the weights of the random RNNs are untied across layers in the tree (TNN). In other words, different random weights are used at each depth of the tree. Since weights are still tied inside each layer this setting can be seen as a convolution where the stride size is equal to the filter size. We call this a tree neural network (TNN) because it is technically not a recursive neural network. While this results in a large increase in parameters, it actually hurts performance underlining the fact that tying the weights in RNNs is beneficial.

**Trained RNN**. Another comparison shown in Fig. 4 (left) is between many random RNNs and a single trained RNN. We carefully cross validated the RNN training procedure, objectives (adding reconstruction costs at each layer as in [10], classifying each layer or only at the top node), regularization, layer size etc. The best performance was still lacking compared to 128 random RNNs ( 2% difference) and training time is much longer. With a more efficient GPU-based implementation it might be possible to train many RNNs in the future.

**Number of random RNNs**: Fig. 4 (center) shows that increasing the number of random RNNs improves performance, leveling off at around 64 on this dataset.

**RGB & depth combinations and features**: Fig. 4 (right) shows that combining RGB and depth features from RNNs improves performance. The two modalities complement each other and produce features that are independent enough so that the classifier can benefit from their combination.

**Global autoencoder on pixels and depth**. In this experiment we investigate whether CNN-RNNs learn better features than simply using a single layer of features on raw pixels. Many methods such as those of Coates and Ng [28] show remarkable results with a single very wide layer. The global autoencoder achieves only 61.1%, (it is overfitting at 93.3% training accuracy). We cross-validated

| Filters | 2nd Layer | Acc. |
|---------|-----------|------|
| See [17] | CNN | 77.66 |
| See [17] | RNN | 77.04 |
| $k$-means | tRNN | 78.10 |
| $k$-means | TNN | 79.67 |
| $k$-means | CNN | 78.65 |
| $k$-means | RNN* | 80.15 |

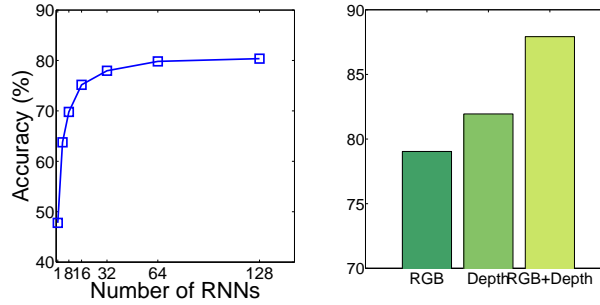

Figure 4: Model analysis on the development split (left and center use rgb only). **Left:** Comparison of two layer CNN with CNN-RNN with different pre-processing ([17] and [13]). TNN is a tree structured neural net with untied weights across layers, tRNN is a single RNN trained with backpropagation (see text for details). The best performance is achieved with our model of random RNNs (marked with ∗). **Center:** Increasing the number of random RNNs improves performance. **Right:** Combining both modalities improves performance to 88% on the development split.

over the number of hidden units and sparsity parameters). This shows that even random recursive neural nets can clearly capture more of the underlying class structure in their feature representations than a single layer autoencoder.

### 4.3   Error Analysis

Fig. 5 shows our confusion matrix across all 51 classes. Most model confusions are very reasonable showing that recursive deep learning methods on raw pixels and depth can give provide high quality features. The only class that we consistently misclassify are mushrooms which are very similar in appearance to garlic.

Fig. 6 shows 4 pairs of often confused classes. Both garlic and mushrooms have very similar appearances and colors. Water bottles and shampoo bottles in particular are problematic because the IR sensors do not properly reflect from see through surfaces.

## 5   Conclusion

We introduced a new model based on a combination of convolutional and recursive neural networks. Unlike previous RNN models, we fix the tree structure, allow multiple vectors to be combined, use multiple RNN weights and keep parameters randomly initialized. This architecture allows for parallelization and high speeds, outperforms two layer CNNs and obtains state of the art performance without any external features. We also demonstrate the applicability of convolutional and recursive feature learning to the new domain of depth images.

### Acknowledgments

We thank Stephen Miller and Alex Teichman for tips on 3D images, Adam Coates for chats about image pre-processing and Ilya Sutskever and Andrew Maas for comments on the paper. We thank the anonymous reviewers for insightful comments. Richard is supported by the Microsoft Research PhD fellowship. The authors gratefully acknowledge the support of the Defense Advanced Research Projects Agency (DARPA) Machine Reading Program under Air Force Research Laboratory (AFRL) prime contract no. FA8750-09-C-0181, and the DARPA Deep Learning program under contract number FA8650-10-C-7020. Any opinions, findings, and conclusions or recommendations expressed in this material are those of the authors and do not necessarily reflect the view of DARPA, AFRL, or the US government.

## References

[1] M. Quigley, S. Batra, S. Gould, E. Klingbeil, Q. Le, A. Wellman, and A.Y. Ng. High-accuracy 3D sensing for mobile manipulation: improving object detection and door opening. In *ICRA*, 2009.

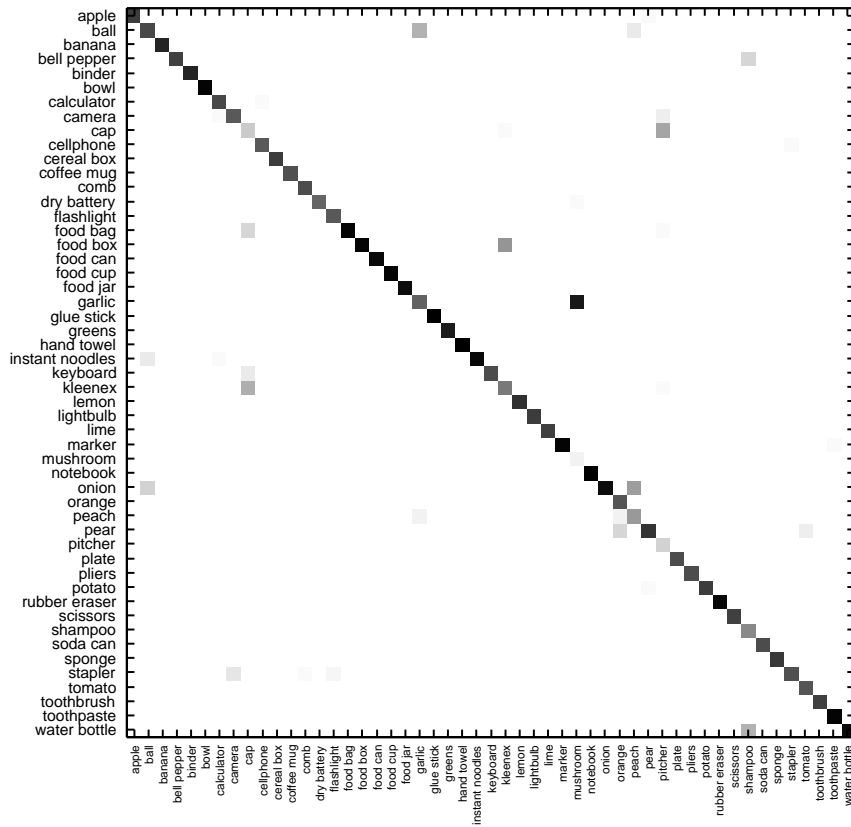

Figure 5: Confusion Matrix of our CNN-RNN model. The ground truth labels are on the y-axis and the predicted labels on the x-axis. Many misclassifications are between (a) garlic and mushroom (b) food-box and kleenex.

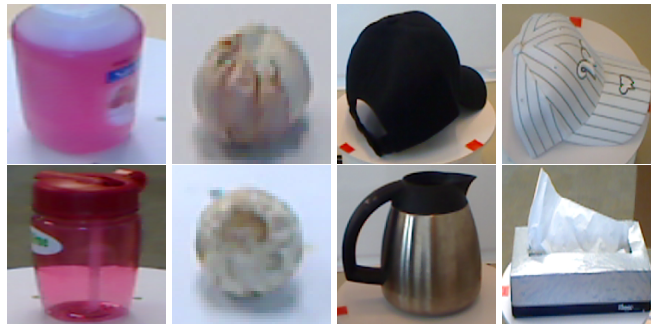

Figure 6: Examples of confused classes: Shampoo bottle and water bottle, mushrooms labeled as garlic, pitchers classified as caps due to shape and color similarity, white caps classified as kleenex boxes at certain angles.

[2] K. Lai, L. Bo, X. Ren, and D. Fox. A Large-Scale Hierarchical Multi-View RGB-D Object Dataset. In *ICRA*, 2011.

[3] A. Johnson. *Spin-Images: A Representation for 3-D Surface Matching*. PhD thesis, Robotics Institute, Carnegie Mellon University, 1997.

[4] H.S. Koppula, A. Anand, T. Joachims, and A. Saxena. Semantic labeling of 3d point clouds for indoor scenes. In *NIPS*, 2011.

[5] L. Bo, X. Ren, and D. Fox. Depth kernel descriptors for object recognition. In *IROS*, 2011.

[6] M. Blum, J. T. Springenberg, J. Wlfing, and M. Riedmiller. A Learned Feature Descriptor for Object Recognition in RGB-D Data. In *ICRA*, 2012.

[7] L. Bo, X. Ren, and D. Fox. Unsupervised Feature Learning for RGB-D Based Object Recognition. In *ISER*, June 2012.

[8] Y. LeCun, L. Bottou, Y. Bengio, and P. Haffner. Gradient-based learning applied to document recognition. *Proceedings of the IEEE*, 86(11), November 1998.

[9] R. Socher, C. Lin, A. Y. Ng, and C.D. Manning. Parsing Natural Scenes and Natural Language with Recursive Neural Networks. In *ICML*, 2011.

[10] R. Socher, J. Pennington, E. H. Huang, A. Y. Ng, and C. D. Manning. Semi-Supervised Recursive Autoencoders for Predicting Sentiment Distributions. In *EMNLP*, 2011.

[11] C. Goller and A. Küchler. Learning task-dependent distributed representations by backpropagation through structure. In *Proceedings of the International Conference on Neural Networks (ICNN-96)*, 1996.

[12] R. Socher, C. D. Manning, and A. Y. Ng. Learning continuous phrase representations and syntactic parsing with recursive neural networks. In *Proceedings of the NIPS-2010 Deep Learning and Unsupervised Feature Learning Workshop*, 2010.

[13] A. Coates, A. Y. Ng, and H. Lee. An Analysis of Single-Layer Networks in Unsupervised Feature Learning. *Journal of Machine Learning Research - Proceedings Track: AISTATS*, 2011.

[14] Q.V. Le, M.A. Ranzato, R. Monga, M. Devin, K. Chen, G.S. Corrado, J. Dean, and A.Y. Ng. Building high-level features using large scale unsupervised learning. In *ICML*, 2012.

[15] Kevin Jarrett, Koray Kavukcuoglu, Marc'Aurelio Ranzato, and Yann LeCun. What is the best multi-stage architecture for object recognition? In *ICCV*, 2009.

[16] A. Saxe, P.W. Koh, Z. Chen, M. Bhand, B. Suresh, and A. Y. Ng. On random weights and unsupervised feature learning. In *ICML*, 2011.

[17] K. Jarrett and K. Kavukcuoglu and M. Ranzato and Y. LeCun. What is the Best Multi-Stage Architecture for Object Recognition? In *ICCV*. IEEE, 2009.

[18] N. Pinto, D. D. Cox, and J. J. DiCarlo. Why is real-world visual object recognition hard? *PLoS Comput Biol*, 2008.

[19] J. B. Pollack. Recursive distributed representations. *Artificial Intelligence*, 46, 1990.

[20] R. Socher, E. H. Huang, J. Pennington, A. Y. Ng, and C. D. Manning. Dynamic Pooling and Unfolding Recursive Autoencoders for Paraphrase Detection. In *NIPS*. MIT Press, 2011.

[21] N. Silberman and R. Fergus. Indoor scene segmentation using a structured light sensor. In *ICCV - Workshop on 3D Representation and Recognition*, 2011.

[22] H. Bay, A. Ess, T. Tuytelaars, and L. Van Gool. Speeded-Up Robust Features (SURF). *Computer Vision and Image Understanding*, 110(3), 2008.

[23] A. E. Abdel-Hakim and A. A. Farag. CSIFT: A SIFT descriptor with color invariant characteristics. In *CVPR*, 2006.

[24] K. Grauman and T. Darrell. The Pyramid Match Kernel: Discriminative Classification with Sets of Image Features. *ICCV*, 2005.

[25] G. Hinton and R. Salakhutdinov. Reducing the dimensionality of data with neural networks. *Science*, 313(5786), 2006.

[26] Y. Bengio. Learning deep architectures for AI. *Foundations and Trends in Machine Learning*, 2(1), 2009.

[27] M. Ranzato, F. J. Huang, Y. Boureau, and Y. LeCun. Unsupervised learning of invariant feature hierarchies with applications to object recognition. *CVPR*, 0:1–8, 2007.

[28] A. Coates and A. Ng. The Importance of Encoding Versus Training with Sparse Coding and Vector Quantization . In *ICML*, 2011.

[29] Farabet C., Couprie C., Najman L., and LeCun Y. Scene parsing with multiscale feature learning, purity trees, and optimal covers. In *ICML*, 2012.

[30] A. Hyvärinen and E. Oja. Independent component analysis: algorithms and applications. *Neural Netw.*, 13, 2000.

[31] J. Ngiam, P. Koh, Z. Chen, S. Bhaskar, and A.Y. Ng. Sparse filtering. In *NIPS*. 2011.

[32] D. C. Ciresan, U. Meier, J. Masci, L. M. Gambardella, and J. Schmidhuber. Flexible, high performance convolutional neural networks for image classification. In *IJCAI*, 2011.

[33] C. Farabet, B. Martini, P. Akselrod, S. Talay, Y. LeCun, and E. Culurciello. Hardware accelerated convolutional neural networks for synthetic vision systems. In *Proc. International Symposium on Circuits and Systems (ISCAS'10)*, 2010.

